# A Linear Programming Approach to Novelty Detection

**Colin Campbell**
Dept. of Engineering Mathematics,
Bristol University, Bristol
Bristol, BS8 1TR,
United Kingdon
*C.Campbell@bris.ac.uk*

**Kristin P. Bennett**
Dept. of Mathematical Sciences
Rensselaer Polytechnic Institute
Troy, New York 12180-3590
United States
*bennek@rpi.edu*

## Abstract

Novelty detection involves modeling the normal behaviour of a system hence enabling detection of any divergence from normality. It has potential applications in many areas such as detection of machine damage or highlighting abnormal features in medical data. One approach is to build a hypothesis estimating the support of the normal data i.e. constructing a function which is positive in the region where the data is located and negative elsewhere. Recently kernel methods have been proposed for estimating the support of a distribution and they have performed well in practice - training involves solution of a quadratic programming problem. In this paper we propose a simpler kernel method for estimating the support based on linear programming. The method is easy to implement and can learn large datasets rapidly. We demonstrate the method on medical and fault detection datasets.

## 1 Introduction.

An important classification task is the ability to distinguish between new instances similar to members of the training set and *all* other instances that can occur. For example, we may want to learn the normal running behaviour of a machine and highlight any significant divergence from normality which may indicate onset of damage or faults. This issue is a generic problem in many fields. For example, an abnormal event or feature in medical diagnostic data typically leads to further investigation.

Novel events can be highlighted by constructing a real-valued density estimation function. However, here we will consider the simpler task of modelling the *support* of a data distribution i.e. creating a binary-valued function which is positive in those regions of input space where the data predominantly lies and negative elsewhere.

Recently kernel methods have been applied to this problem [4]. In this approach data is implicitly mapped to a high-dimensional space called *feature space* [13]. Suppose the data points in *input space* are $\mathbf{x}_i$ (with $i = 1, \ldots, m$) and the mapping

is $\mathbf{x}_i \to \phi(\mathbf{x}_i)$ then in the span of $\{\phi(\mathbf{x}_i)\}$, we can expand a vector $\mathbf{w} = \sum_j \alpha_j \phi(\mathbf{x}_j)$. Hence we can define separating hyperplanes in feature space by $\mathbf{w} \cdot \phi(\mathbf{x}_i) + b = 0$. We will refer to $\mathbf{w} \cdot \phi(\mathbf{x}_i) + b$ as the *margin* which will be positive on one side of the separating hyperplane and negative on the other. Thus we can also define a decision function:

$$ sign\left(\mathbf{w} \cdot \phi(\mathbf{z}) + b\right) = sign\left(\sum_j \alpha_j \phi(\mathbf{x}_j) \cdot \phi(\mathbf{z}) + b\right) \tag{1} $$

where $\mathbf{z}$ is a new data point. The data appears in the form of an inner product in feature space so we can implicitly define feature space by our choice of *kernel* function:

$$ K(\mathbf{x}_i, \mathbf{x}_j) = \phi(\mathbf{x}_i) \cdot \phi(\mathbf{x}_j) \tag{2} $$

A number of choices for the kernel are possible, for example, RBF kernels:

$$ K(\mathbf{x}_i, \mathbf{x}_j) = e^{-\|\mathbf{x}_i - \mathbf{x}_j\|^2 / 2\sigma^2} \tag{3} $$

With the given kernel the decision function is therefore given by:

$$ sign\left(\sum_j \alpha_j K(\mathbf{x}_j, \mathbf{z}) + b\right) \tag{4} $$

One approach to novelty detection is to find a hypersphere in feature space with a minimal radius $R$ and centre $\mathbf{a}$ which contains most of the data: novel test points lie outside the boundary of this hypersphere [3, 12]. This approach to novelty detection was proposed by Tax and Duin [10] and successfully used on real life applications [11]. The effect of outliers is reduced by using slack variables $\xi_i$ to allow for datapoints outside the sphere and the task is to minimise the volume of the sphere and number of datapoints outside i.e.

$$ \begin{array}{ll} \min & \left[R^2 + \lambda \sum_i \xi_i\right] \\ s.t. & (\mathbf{x}_i - \mathbf{a}) \cdot (\mathbf{x}_i - \mathbf{a}) \leq R^2 + \xi_i, \quad \xi_i \geq 0 \end{array} \tag{5} $$

Since the data appears in the form of inner products kernel substitution can be applied and the learning task can be reduced to a quadratic programming problem. An alternative approach has been developed by Schölkopf et al. [7]. Suppose we restricted our attention to RBF kernels (3) then the data lies on the surface of a hypersphere in feature space since $\phi(\mathbf{x}) \cdot \phi(\mathbf{x}) = K(\mathbf{x}, \mathbf{x}) = 1$. The objective is therefore to separate off the surface region constaining data from the region containing no data. This is achieved by constructing a hyperplane which is maximally distant from the origin with all datapoints lying on the opposite side from the origin and such that the margin is positive. The learning task in dual form involves minimisation of:

$$ \begin{array}{ll} \min & W(\alpha) = \frac{1}{2} \sum_{i,k=1}^m \alpha_i \alpha_j K(\mathbf{x}_i, \mathbf{x}_j) \\ s.t. & 0 \leq \alpha_i \leq C, \quad \sum_{i=1}^m \alpha_i = 1. \end{array} \tag{6} $$

However, the origin plays a special role in this model. As the authors point out [9] this is a disadvantage since the origin effectively acts as a prior for where the class of abnormal instances is assumed to lie. In this paper we avoid this problem: rather than *repelling* the hyperplane away from an arbitrary point outside the data distribution we instead try and *attract* the hyperplane towards the centre of the data distribution.

In this paper we will outline a new algorithm for novelty detection which can be easily implemented using linear programming (LP) techniques. As we illustrate in section 3 it performs well in practice on datasets involving the detection of abnormalities in medical data and fault detection in condition monitoring.

## 2   The Algorithm

For the hard margin case (see Figure 1) the objective is to find a surface in input space which wraps around the data clusters: anything outside this surface is viewed as abnormal. This surface is defined as the level set, $f(\mathbf{z}) = 0$, of some nonlinear function. In feature space, $f(\mathbf{z}) = \sum_i \alpha_i K(\mathbf{z}, \mathbf{x}_i) + b$, this corresponds to a hyperplane which is pulled onto the mapped datapoints with the restriction that the margin always remains positive or zero. We make the fit of this nonlinear function or hyperplane as tight as possible by minimizing the mean value of the output of the function, i.e., $\sum_i f(\mathbf{x}_i)$. This is achieved by minimising:

$$W(\alpha, b) = \sum_{i=1}^{m} \left( \sum_{j=1}^{m} \alpha_j K(\mathbf{x}_i, \mathbf{x}_j) + b \right) \qquad (7)$$

subject to:

$$\sum_{j=1}^{m} \alpha_j K(\mathbf{x}_i, \mathbf{x}_j) + b \geq 0 \qquad (8)$$

$$\sum_{i=1}^{m} \alpha_i = 1, \quad \alpha_i \geq 0 \qquad (9)$$

The bias $b$ is just treated as an additional parameter in the minimisation process though unrestricted in sign. The added constraints (9) on $\alpha$ bound the class of models to be considered - we don't want to consider simple linear rescalings of the model. These constraints amount to a choice of scale for the weight vector normal to the hyperplane in feature space and hence do not impose a restriction on the model. Also, these constraints ensure that the problem is well-posed and that an optimal solution with $\alpha \neq 0$ exists. Other constraints on the class of functions are possible, e.g. $\|\alpha\|_1 = 1$ with no restriction on the sign of $\alpha_i$.

Many real-life datasets contain noise and outliers. To handle these we can introduce a *soft margin* in analogy to the usual approach used with support vector machines. In this case we minimise:

$$W(\alpha, b) = \sum_{i=1}^{m} \left( \sum_{j=1}^{m} \alpha_j K(\mathbf{x}_i, \mathbf{x}_j) + b \right) + \lambda \sum_{i=1}^{m} \xi_i \qquad (10)$$

subject to:

$$\sum_{j=1}^{m} \alpha_j K(\mathbf{x}_i, \mathbf{x}_j) + b \geq -\xi_i, \quad \xi_i \geq 0 \tag{11}$$

and constraints (9). The parameter $\lambda$ controls the extent of margin errors (larger $\lambda$ means fewer outliers are ignored: $\lambda \rightarrow \infty$ corresponds to the *hard margin* limit).

The above problem can be easily solved for problems with thousands of points using standard simplex or interior point algorithms for linear programming. With the addition of column generation techniques, these same approaches can be adopted for very large problems in which the kernel matrix exceeds the capacity of main memory. Column generation algorithms incrementally add and drop columns each corresponding to a single kernel function until optimality is reached. Such approaches have been successfully applied to other support vector problems [6, 2]. Basic simplex algorithms were sufficient for the problems considered in this paper, so we defer a listing of the code for column generation to a later paper together with experiments on large datasets [1].

## 3 Experiments

**Artificial datasets.** Before considering experiments on real-life data we will first illustrate the performance of the algorithm on some artificial datasets. In Figure 1 the algorithm places a boundary around two data clusters in input space: a hard margin was used with RBF kernels and $\sigma = 0.2$. In Figure 2 four outliers lying outside a single cluster are ignored when the system is trained using a soft margin. In Figure 3 we show the effect of using a modified RBF kernel $K(\mathbf{x}_i, \mathbf{x}_j) = e^{-|\mathbf{x}_i - \mathbf{x}_j|/2\sigma^2}$. This kernel and the one in (3) use a measure $\mathbf{x} - \mathbf{y}$, thus $K(\mathbf{x}, \mathbf{x})$ is constant and the points lie on the surface of a hypersphere in feature space. As a consequence a hyperplane slicing through this hypersphere gives a closed boundary separating normal and abnormal in input space: however, we found other choices of kernels may not produce closed boundaries in input space.

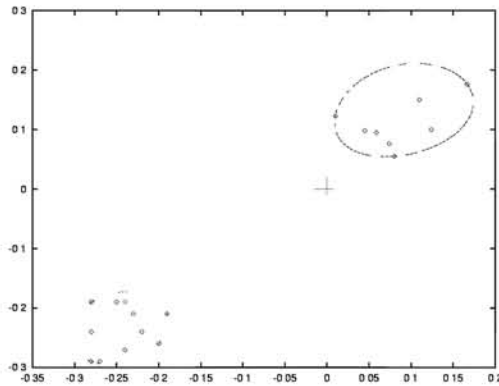

Figure 1: The solution in input space for the hyperplane minimising $W(\alpha, b)$ in equation (7). A hard margin was used with RBF kernels trained using $\sigma = 0.2$

**Medical Diagnosis.** For detection of abnormalities in medical data we investigated performance on the *Biomed* dataset [5] from the Statlib data archive [14].

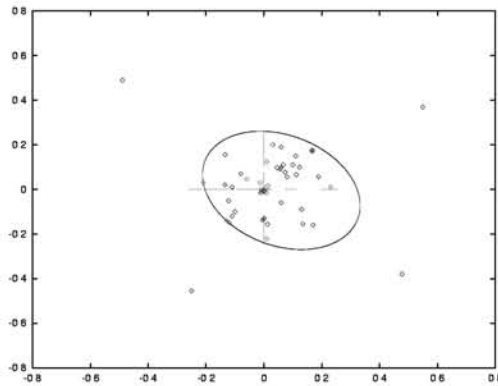

Figure 2: In this example 4 outliers are ignored by using a soft margin (with $\lambda = 10.0$). RBF kernels were used with $\sigma = 0.2$

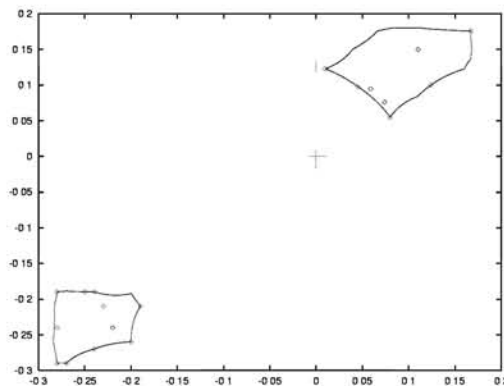

Figure 3: The solution in input space for a modified RBF kernel $K(\mathbf{x}_i, \mathbf{x}_j) = e^{-|\mathbf{x}_i - \mathbf{x}_j|/2\sigma^2}$ with $\sigma = 0.5$

This dataset consisted of 194 observations each with 4 attributes corresponding to measurements made on blood samples (15 observations with missing values were removed). We trained the system on 100 randomly chosen normal observations from healthy patients. The system was then tested on 27 normal observations and 67 observations which exhibited abnormalities due to the presense of a rare genetic disease.

In Figure 4 we plot the results for training the novelty detector using a hard margin and with RBF kernels. This plot gives the error rate (as a percentage) on the $y$-axis, versus $\sigma$ on the $x$-axis with the solid curve giving the performance on normal observations in the test data and the dashed curve giving performance on abnormal observations. Clearly, when $\sigma$ is very small the system puts a Gaussian of narrow width around each data point and hence all test data is labelled as abnormal. As $\sigma$ increases the model improves and at $\sigma = 1.1$ all but 2 of the normal test observations are correctly labelled and 57 of the 67 abnormal observations are correctly labelled. As $\sigma$ increases to $\sigma = 10.0$ the solution has 1 normal test observation incorrectly labelled and 29 abnormal observations correctly labelled.

The kernel parameter $\sigma$ is therefore crucial is determining the balance between

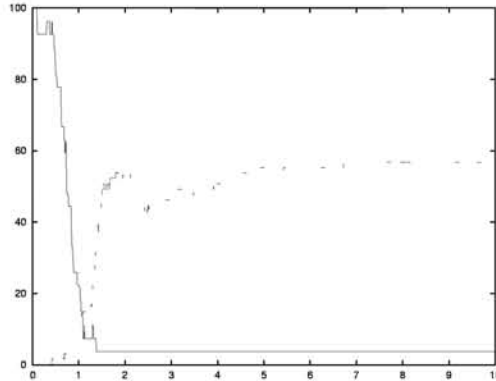

Figure 4: The error rate (as a percentage) on the $y$-axis, versus $\sigma$ on the $x$-axis. The solid curve giving the performance on normal observations in the test data and the dashed curve giving performance on abnormal observations.

normality and abnormality. Future research on model selection may indicate a good choice for the kernel parameter. However, if the dataset is large enough and some abnormal events are known then a validation study can be used to determine the kernel parameter - as we illustrate with the application below. Interestingly, if we use an ensemble of models instead, with $\sigma$ chosen across a range, then the relative proportion indicating abnormality gives an approximate measure of the confidence in the novelty of an observation: 29 observations are abnormal for all $\sigma$ in Figure 4 and hence must be abnormal with high confidence.

**Condition Monitoring**. Fault detection is an important generic problem in the condition monitoring of machinery: failure to detect faults can lead to machine damage, while an oversensitive fault detection system can lead to expensive and unnecessary downtime. An an example we will consider detection of 4 classes of fault in ball-bearing cages, which are often safety critical components in machines, vehicles and other systems such as aircraft wing flaps.

In this study we used a dataset from the Structural Integrity and Damage Assessment Network [15]. Each *instance* consisted of 2048 samples of acceleration taken with a Bruel and Kjaer vibration analyser. After pre-processing with a discrete Fast Fourier Transform each such instance had 32 attributes characterising the measured signals.

The dataset consisted of 5 categories: normal data corresponding to measurements made from new ball-bearings and 4 types of abnormalities which we will call *type 1* (outer race completely broken), *type 2* (broken cage with one loose element), *type 3* (damaged cage with four loose elements) and *type 4* (a badly worn ball-bearing with no evident damage). To train the system we used 913 normal instances on new ball-bearings. Using RBF kernels the best value of $\sigma$ ($\sigma = 320.0$) was found using a validation study consisting of 913 new normal instances, 747 instances of type 1 faults and 996 instances of type 2 faults. On new test data 98.7% of normal instances were correctly labelled (913 instances), 100% of type 1 instances were correctly labelled (747 instances) and 53.3% of type 2 instances were correctly labelled (996 instances). Of course, with ample normal and abnormal data this problem could also be approached using a binary classifier instead. Thus to evaluate performance on totally unseen abnormalities we tested the novelty detector on type 3 errors and type 4 errors (with 996 instances of each). The novelty detector labelled 28.3% of

type 3 and 25.5% of type 4 instances as abnormal - which was statistically significant against a background of 1.3% errors on normal data.

## 4    Conclusion

In this paper we have presented a new kernelised novelty detection algorithm which uses linear programming techniques rather than quadratic programming. The algorithm is simple, easy to implement with standard LP software packages and it performs well in practice. The algorithm is also very fast in execution: for the 913 training examples used in the experiments on condition monitoring the model was constructed in about 4 seconds using a Silicon Graphics Origin 200.

## References

[1] K. Bennett and C. Campbell. A Column Generation Algorithm for Novelty Detection. Preprint in preparation.

[2] K. Bennett, A. Demiriz and J. Shawe-Taylor, A Column Generation Algorithm for Boosting. In *Proceed. of Intl. Conf. on Machine Learning*. Stanford, CA, 2000.

[3] C. Burges. A tutorial on support vector machines for pattern recognition. *Data Mining and Knowledge Discovery*, 2, p. 121-167, 1998.

[4] C. Campbell. An Introduction to Kernel Methods. In: *Radial Basis Function Networks: Design and Applications*. R.J. Howlett and L.C. Jain (eds). Physica Verlag, Berlin, to appear.

[5] L. Cox, M. Johnson and K. Kafadar. Exposition of Statistical Graphics Technology. *ASA Proceedings of the Statistical Computation Section*, p. 55-56, 1982.

[6] O. L. Mangasarian and D. Musicant. Massive Support Vector Regression. Data Mining Institute Technical Report 99-02, University of Wisconsin-Madison, 1999.

[7] B. Schölkopf, J.C. Platt, J. Shawe-Taylor, A.J. Smola, R.C. Williamson. Estimating the support of a high-dimensional distribution. Microsoft Research Corporation Technical Report MSR-TR-99-87, 1999, 2000

[8] B. Schölkopf, R. Williamson, A. Smola, and J. Shawe-Taylor. SV estimation of a distribution's support. In *Neural Information Processing Systems*, 2000, to appear.

[9] B. Schölkopf, J. Platt and A. Smola. Kernel Method for Percentile Feature Extraction. Microsoft Technical Report MSR-TR-2000-22.

[10] D. Tax and R. Duin. Data domain description by Support Vectors. In *Proceedings of ESANN99*, ed. M Verleysen, D. Facto Press, Brussels, p. 251-256, 1999.

[11] D. Tax, A. Ypma, and R. Duin. Support vector data description applied to machine vibration analysis. In: M. Boasson, J. Kaandorp, J.Tonino, M. Vosselman (eds.), *Proc. 5th Annual Conference of the Advanced School for Computing and Imaging* (Heijen, NL, June 15-17), 1999, 398-405.

[12] V. Vapnik. *The Nature of Statistical Learning Theory*. Springer, N.Y., 1995.

[13] V. Vapnik. *Statistical Learning Theory*. Wiley, 1998.

[14] cf. http://lib.stat.cmu.edu/datasets

[15] http://www.sidanet.org
